# A Brain-Machine Interface Operating with a Real-Time Spiking Neural Network Control Algorithm

**Julie Dethier**[*]
Department of Bioengineering
Stanford University, CA 94305
jdethier@stanford.edu

**Paul Nuyujukian**
Department of Bioengineering
School of Medicine
Stanford University, CA 94305
paul@npl.stanford.edu

**Chris Eliasmith**
Centre for Theoretical Neuroscience
University of Waterloo, Canada
celiasmith@uwaterloo.ca

**Terry Stewart**
Centre for Theoretical Neuroscience
University of Waterloo, Canada
tcstewar@uwaterloo.ca

**Shauki A. Elassaad**
Department of Bioengineering
Stanford University, CA 94305
shauki@stanford.edu

**Krishna V. Shenoy**
Department of Electrical Engineering
Department of Bioengineering
Department of Neurobiology
Stanford University, CA 94305
shenoy@stanford.edu

**Kwabena Boahen**
Department of Bioengineering
Stanford University, CA 94305
boahen@stanford.edu

## Abstract

Motor prostheses aim to restore function to disabled patients. Despite compelling proof of concept systems, barriers to clinical translation remain. One challenge is to develop a low-power, fully-implantable system that dissipates only minimal power so as not to damage tissue. To this end, we implemented a Kalman-filter based decoder via a spiking neural network (SNN) and tested it in brain-machine interface (BMI) experiments with a rhesus monkey. The Kalman filter was trained to predict the arm's velocity and mapped on to the SNN using the Neural Engineering Framework (NEF). A 2,000-neuron embedded Matlab SNN implementation runs in real-time and its closed-loop performance is quite comparable to that of the standard Kalman filter. The success of this closed-loop decoder holds promise for hardware SNN implementations of statistical signal processing algorithms on neuromorphic chips, which may offer power savings necessary to overcome a major obstacle to the successful clinical translation of neural motor prostheses.

---

[*]Present: Research Fellow F.R.S.-FNRS, Systmod Unit, University of Liege, Belgium.

# 1 Cortically-controlled motor prostheses: the challenge

Motor prostheses aim to restore function for severely disabled patients by translating neural signals from the brain into useful control signals for prosthetic limbs or computer cursors. Several proof of concept demonstrations have shown encouraging results, but barriers to clinical translation still remain. One example is the development of a fully-implantable system that meets power dissipation constraints, but is still powerful enough to perform complex operations. A recently reported closed-loop cortically-controlled motor prosthesis is capable of producing quick, accurate, and robust computer cursor movements by decoding neural signals (threshold-crossings) from a 96-electrode array in rhesus macaque premotor/motor cortex [1]-[4]. This, and previous designs (e.g., [5]), employ versions of the Kalman filter, ubiquitous in statistical signal processing. Such a filter and its variants are the state-of-the-art decoder for brain-machine interfaces (BMIs) in humans [5] and monkeys [2].

While these recent advances are encouraging, clinical translation of such BMIs requires fully-implanted systems, which in turn impose severe power dissipation constraints. Even though it is an open, actively-debated question as to how much of the neural prosthetic system must be implanted, we note that there are no reports to date demonstrating a fully implantable 100-channel wireless transmission system, motivating performing decoding within the implanted chip. This computation is constrained by a stringent power budget: A $6 \times 6\text{mm}^2$ implant must dissipate less than 10mW to avoid heating the brain by more than $1^\circ$C [6], which is believed to be important for long term cell health. With this power budget, current approaches can not scale to higher electrode densities or to substantially more computer-intensive decode/control algorithms.

The feasibility of mapping a Kalman-filter based decoder algorithm [1]-[4] on to a spiking neural network (SNN) has been explored off-line (open-loop). In these off-line tests, the SNN's performance virtually matched that of the standard implementation [7]. These simulations provide confidence that this algorithm—and others similar to it—could be implemented using an ultra-low-power approach potentially capable of meeting the severe power constraints set by clinical translation. This neuromorphic approach uses very-large-scale integrated systems containing microelectronic analog circuits to *morph* neural systems into silicon chips [8, 9]. These neuromorphic circuits may yield tremendous power savings—50nW per silicon neuron [10]—over digital circuits because they use physical operations to perform mathematical computations (analog approach). When implemented on a chip designed using the neuromorphic approach, a 2,000-neuron SNN network can consume as little as $100\mu$W.

Demonstrating this approach's feasibility in a closed-loop system running in real-time is a key, non-incremental step in the development of a fully implantable decoding chip, and is necessary before proceeding with fabricating and implanting the chip. As noise, delay, and over-fitting play a more important role in the closed-loop setting, it is not obvious that the SNN's stellar open-loop performance will hold up. In addition, performance criteria are different in the closed-loop and open-loop settings (e.g., time per target vs. root mean squared error). Therefore, a SNN of a different size may be required to meet the desired specifications. Here we present results and assess the performance and viability of the SNN Kalman-filter based decoder in real-time, closed-loop tests, with the monkey performing a center-out-and-back target acquisition task. To achieve closed-loop operation, we developed an embedded Matlab implementation that ran a 2,000-neuron version of the SNN in real-time on a PC. We achieved almost a 50-fold speed-up by performing part of the computation in a lower-dimensional space defined by the formal method we used to map the Kalman filter on to the SNN. This shortcut allowed us to run a larger SNN in real-time than would otherwise be possible.

# 2 Spiking neural network mapping of control theory algorithms

As reported in [11], a formal methodology, called the *Neural Engineering Framework* (NEF), has been developed to map control-theory algorithms onto a computational fabric consisting of a highly heterogeneous population of spiking neurons simply by programming the strengths of their connections. These artificial neurons are characterized by a nonlinear multi-dimensional-vector-to-spike-rate function—$a_i(\mathbf{x}(t))$ for the $i^{\text{th}}$ neuron—with parameters (preferred direction, maximum firing rate, and spiking-threshold) drawn randomly from a wide distribution (standard deviation $\approx$ mean).

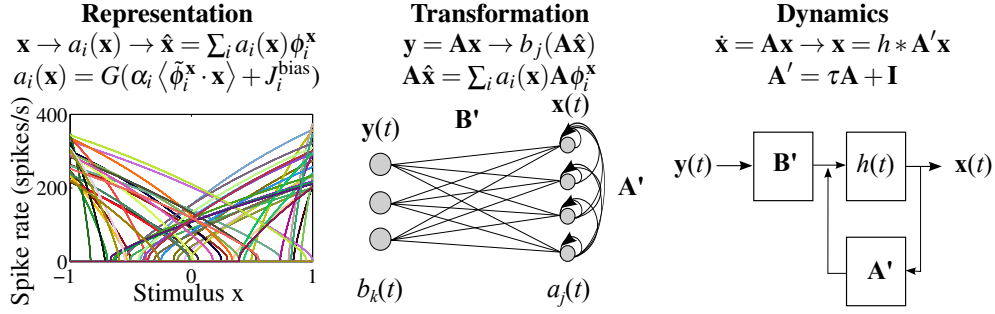

Figure 1: NEF's three principles. **Representation.** 1D tuning curves of a population of 50 leaky integrate-and-fire neurons. The neurons' tuning curves map control variables ($\mathbf{x}$) to spike rates ($a_i(\mathbf{x})$); this nonlinear transformation is inverted by linear weighted decoding. $G()$ is the neurons' nonlinear current-to-spike-rate function. **Transformation.** SNN with populations $b_k(t)$ and $a_j(t)$ representing $\mathbf{y}(t)$ and $\mathbf{x}(t)$. Feedforward and recurrent weights are determined by $\mathbf{B}'$ and $\mathbf{A}'$, as described next. **Dynamics.** The system's dynamics is captured in a neurally plausible fashion by replacing integration with the synapses' spike response, $h(t)$, and replacing the matrices with $\mathbf{A}' = \tau\mathbf{A} + \mathbf{I}$ and $\mathbf{B}' = \tau\mathbf{B}$ to compensate.

The neural engineering approach to configuring SNNs to perform arbitrary computations is underlined by three principles (Figure 1) [11]-[14]:

**Representation** is defined by nonlinear encoding of $\mathbf{x}(t)$ as a spike rate, $a_i(\mathbf{x}(t))$—represented by the *neuron tuning curve*—combined with optimal weighted linear decoding of $a_i(\mathbf{x}(t))$ to recover an estimate of $\mathbf{x}(t)$, $\hat{\mathbf{x}}(t) = \sum_i a_i(\mathbf{x}(t))\phi_i^{\mathbf{x}}$, where $\phi_i^{\mathbf{x}}$ are the decoding weights.

**Transformation** is performed by using alternate decoding weights in the decoding operation to map transformations of $\mathbf{x}(t)$ directly into transformations of $a_i(\mathbf{x}(t))$. For example, $\mathbf{y}(t) = \mathbf{A}\mathbf{x}(t)$ is represented by the spike rates $b_j(\mathbf{A}\hat{\mathbf{x}}(t))$, where unit $j$'s input is computed directly from unit $i$'s output using $\mathbf{A}\hat{\mathbf{x}}(t) = \sum_i a_i(\mathbf{x}(t))\mathbf{A}\phi_i^{\mathbf{x}}$, an alternative linear weighting.

**Dynamics** brings the first two principles together and adds the time dimension to the circuit. This principle aims at reuniting the control-theory and neural levels by modifying the matrices to render the system *neurally plausible*, thereby permitting the synapses' spike response, $h(t)$, (i.e., impulse response) to capture the system's dynamics. For example, for $h(t) = \tau^{-1}e^{-t/\tau}$, $\dot{\mathbf{x}} = \mathbf{A}\mathbf{x}(t)$ is realized by replacing $\mathbf{A}$ with $\mathbf{A}' = \tau\mathbf{A} + \mathbf{I}$. This so-called *neurally plausible matrix* yields an equivalent dynamical system: $\mathbf{x}(t) = h(t) * \mathbf{A}'\mathbf{x}(t)$, where convolution replaces integration.

The nonlinear encoding process—from a multi-dimensional stimulus, $\mathbf{x}(t)$, to a one-dimensional soma current, $J_i(\mathbf{x}(t))$, to a firing rate, $a_i(\mathbf{x}(t))$—is specified as:

$$a_i(\mathbf{x}(t)) = G(J_i(\mathbf{x}(t))). \tag{1}$$

Here $G$ is the neurons' nonlinear current-to-spike-rate function, which is given by

$$G(J_i(\mathbf{x})) = \left\{ \tau^{\text{ref}} - \tau^{\text{RC}}\ln\left(1 - J_{\text{th}}/J_i(\mathbf{x})\right) \right\}^{-1}, \tag{2}$$

for the leaky integrate-and-fire model (LIF). The LIF neuron has two behavioral regimes: sub-threshold and super-threshold. The sub-threshold regime is described by an RC circuit with time constant $\tau^{\text{RC}}$. When the sub-threshold soma voltage reaches the threshold, $V_{\text{th}}$, the neuron emits a spike $\delta(t - t_n)$. After this spike, the neuron is reset and rests for $\tau^{\text{ref}}$ seconds (absolute refractory period) before it resumes integrating. $J_{\text{th}} = V_{\text{th}}/R$ is the minimum input current that produces spiking. Ignoring the soma's RC time-constant when specifying the SNN's dynamics are reasonable because the neurons cross threshold at a rate that is proportional to their input current, which thus sets the spike rate instantaneously, without any filtering [11].

The conversion from a multi-dimensional stimulus, $\mathbf{x}(t)$, to a one-dimensional soma current, $J_i$, is performed by assigning to the neuron a preferred direction, $\tilde{\phi}_i^{\mathbf{x}}$, in the stimulus space and taking the dot-product:

$$J_i(\mathbf{x}(t)) = \alpha_i \left\langle \tilde{\phi}_i^{\mathbf{x}} \cdot \mathbf{x}(t) \right\rangle + J_i^{\text{bias}}, \tag{3}$$

where $\alpha_i$ is a gain or conversion factor, and $J_i^{\text{bias}}$ is a bias current that accounts for background activity. For a 1D space, $\tilde{\phi}_i^{\mathbf{x}}$ is either $+1$ or $-1$ (drawn randomly), for ON and OFF neurons, respectively. The resulting *tuning curves* are illustrated in Figure 1, *left*.

The linear decoding process is characterized by the synapses' spike response, $h(t)$ (i.e., post-synaptic currents), and the decoding weights, $\phi_i^{\mathbf{x}}$, which are obtained by minimizing the mean square error. A single noise term, $\eta$, takes into account all sources of noise, which have the effect of introducing uncertainty into the decoding process. Hence, the transmitted firing rate can be written as $a_i(\mathbf{x}(t)) + \eta_i$, where $a_i(\mathbf{x}(t))$ represents the noiseless set of tuning curves and $\eta_i$ is a random variable picked from a zero-mean Gaussian distribution with variance $\sigma^2$. Consequently, the mean square error can be written as [11]:

$$ E \quad = \quad \frac{1}{2}\left\langle [\mathbf{x}(t) - \hat{\mathbf{x}}(t)]^2 \right\rangle_{\mathbf{x},\eta,t} = \frac{1}{2}\left\langle \left[ \mathbf{x}(t) - \sum_i (a_i(\mathbf{x}(t)) + \eta_i)\,\phi_i^{\mathbf{x}} \right]^2 \right\rangle_{\mathbf{x},\eta,t} \tag{4} $$

where $\langle \cdot \rangle_{\mathbf{x},\eta}$ denotes integration over the range of $\mathbf{x}$ and $\eta$, the expected noise. We assume that the noise is independent and has the same variance for each neuron [11], which yields:

$$ E = \frac{1}{2}\left\langle \left[ \mathbf{x}(t) - \sum_i a_i(\mathbf{x}(t))\phi_i^{\mathbf{x}} \right]^2 \right\rangle_{\mathbf{x},t} + \frac{1}{2}\sigma^2 \sum_i (\phi_i^{\mathbf{x}})^2, \tag{5} $$

where $\sigma^2$ is the noise variance $\eta_i \eta_j$. This expression is minimized by:

$$ \phi_i^{\mathbf{x}} = \sum_j^N \Gamma_{ij}^{-1} \Upsilon_j, \tag{6} $$

with $\Gamma_{ij} = \left\langle a_i(\mathbf{x})a_j(\mathbf{x}) \right\rangle_{\mathbf{x}} + \sigma^2 \delta_{ij}$, where $\delta$ is the Kronecker delta function matrix, and $\Upsilon_j = \left\langle \mathbf{x}a_j(\mathbf{x}) \right\rangle_{\mathbf{x}}$ [11]. One consequence of modeling noise in the neural representation is that the matrix $\Gamma$ is invertible despite the use of a highly overcomplete representation. In a noiseless representation, $\Gamma$ is generally singular because, due to the large number of neurons, there is a high probability of having two neurons with similar tuning curves leading to two similar rows in $\Gamma$.

## 3   Kalman-filter based cortical decoder

In the 1960's, Kalman described a method that uses linear filtering to track the state of a dynamical system throughout time using a model of the dynamics of the system as well as noisy measurements [15]. The model dynamics gives an estimate of the state of the system at the next time step. This estimate is then corrected using the observations (i.e., measurements) at this time step. The relative weights for these two pieces of information are given by the *Kalman gain*, $\mathbf{K}$ [15, 16]. Whereas the Kalman gain is updated at each iteration, the state and observation matrices (defined below)—and corresponding noise matrices—are supposed constant.

In the case of prosthetic applications, the system's state vector is the cursor's kinematics, $\mathbf{x}_t = [vel_t^x, vel_t^y, 1]$, where the constant 1 allows for a fixed offset compensation. The measurement vector, $\mathbf{y}_t$, is the neural spike rate (spike counts in each time step) of 192 channels of neural threshold crossings. The system's dynamics is modeled by:

$$ \mathbf{x}_t \quad = \quad \mathbf{A}\mathbf{x}_{t-1} + \mathbf{w}_t, \tag{7} $$

$$ \mathbf{y}_t \quad = \quad \mathbf{C}\mathbf{x}_t + \mathbf{q}_t, \tag{8} $$

where $\mathbf{A}$ is the state matrix, $\mathbf{C}$ is the observation matrix, and $\mathbf{w}_t$ and $\mathbf{q}_t$ are additive, Gaussian noise sources with $\mathbf{w}_t \sim \mathcal{N}(0, \mathbf{W})$ and $\mathbf{q}_t \sim \mathcal{N}(0, \mathbf{Q})$. The model parameters ($\mathbf{A}$, $\mathbf{C}$, $\mathbf{W}$ and $\mathbf{Q}$) are fit with training data by correlating the observed hand kinematics with the simultaneously measured neural signals (Figure 2).

For an efficient decoding, we derived the steady-state update equation by replacing the adaptive Kalman gain by its steady-state formulation: $\mathbf{K} = (\mathbf{I} + \mathbf{W}\mathbf{C}\mathbf{Q}^{-1}\mathbf{C})^{-1}\,\mathbf{W}\,\mathbf{C}^{\mathrm{T}}\mathbf{Q}^{-1}$. This yields the following estimate of the system's state:

$$ \mathbf{x}_t = (\mathbf{I} - \mathbf{K}\mathbf{C})\mathbf{A}\mathbf{x}_{t-1} + \mathbf{K}\mathbf{y}_t = \mathbf{M}_x^{\mathrm{DT}}\mathbf{x}_{t-1} + \mathbf{M}_y^{\mathrm{DT}}\mathbf{y}_t, \tag{9} $$

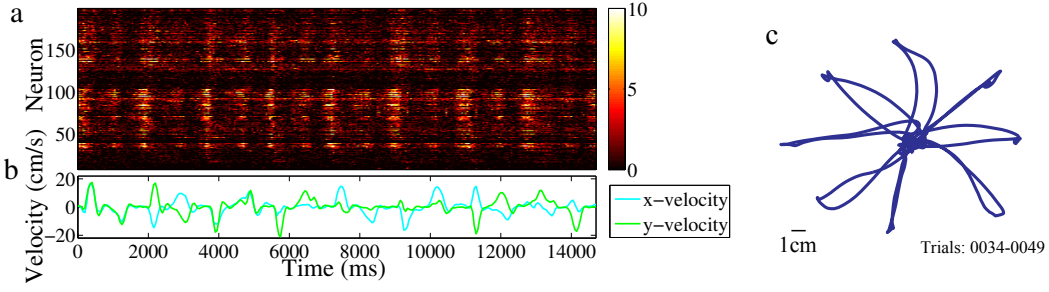

Figure 2: Neural and kinematic measurements (monkey J, 2011-04-16, 16 continuous trials) used to fit the standard Kalman filter model. **a.** The 192 cortical recordings fed as input to fit the Kalman filter's matrices (color code refers to the number of threshold crossings observed in each 50ms bin). **b.** Hand $x$- and $y$-velocity measurements correlated with the neural data to obtain the Kalman filter's matrices. **c.** Cursor kinematics of 16 continuous trials under direct hand control.

where $\mathbf{M}_x^{\text{DT}} = (\mathbf{I} - \mathbf{KC})\mathbf{A}$ and $\mathbf{M}_y^{\text{DT}} = \mathbf{K}$ are the discrete time (DT) Kalman matrices. The steady-state formulation improves efficiency with little loss in accuracy because the optimal Kalman gain rapidly converges (typically less than 100 iterations). Indeed, in neural applications under both open-loop and closed-loop conditions, the difference between the full Kalman filter and its steady-state implementation falls to within 1% in a few seconds [17]. This simplifying assumption reduces the execution time for decoding a typical neuronal firing rate signal approximately seven-fold [17], a critical speed-up for real-time applications.

## 4   Kalman filter with a spiking neural network

To implement the Kalman filter with a SNN by applying the NEF, we first convert Equation 9 from DT to continuous time (CT), and then replace the CT matrices with neurally plausible ones, which yields:

$$\mathbf{x}(t) = h(t) * \left( \mathbf{A}'\mathbf{x}(t) + \mathbf{B}'\mathbf{y}(t) \right), \tag{10}$$

where $\mathbf{A}' = \tau\mathbf{M}_x^{\text{CT}} + \mathbf{I}$, $\mathbf{B}' = \tau\mathbf{M}_y^{\text{CT}}$, with $\mathbf{M}_x^{\text{CT}} = \left(\mathbf{M}_x^{\text{DT}} - \mathbf{I}\right)/\Delta t$ and $\mathbf{M}_y^{\text{CT}} = \left(\mathbf{M}_y^{\text{DT}}\right)/\Delta t$, the CT Kalman matrices, and $\Delta t = 50$ms, the discrete time step; $\tau$ is the synaptic time-constant.

The $j^{\text{th}}$ neuron's input current (see Equation 3) is computed from the system's current state, $\mathbf{x}(t)$, which is computed from estimates of the system's previous state ($\hat{\mathbf{x}}(t) = \sum_i a_i(t)\phi_i^{\mathbf{x}}$) and current input ($\hat{\mathbf{y}}(t) = \sum_k b_k(t)\phi_k^{\mathbf{y}}$) using Equation 10. This yields:

$$\begin{aligned} J_j(\mathbf{x}(t)) &= \alpha_j \left\langle \tilde{\phi}_j^{\mathbf{x}} \cdot \mathbf{x}(t) \right\rangle + J_j^{\text{bias}} \\ &= \alpha_j \left\langle \tilde{\phi}_j^{\mathbf{x}} \cdot h(t) * \left( \mathbf{A}'\hat{\mathbf{x}}(t) + \mathbf{B}'\hat{\mathbf{y}}(t) \right) \right\rangle + J_j^{\text{bias}} \\ &= \alpha_j \left\langle \tilde{\phi}_j^{\mathbf{x}} \cdot h(t) * \left( \mathbf{A}' \sum_i a_i(t)\phi_i^{\mathbf{x}} + \mathbf{B}' \sum_k b_k(t)\phi_k^{\mathbf{y}} \right) \right\rangle + J_j^{\text{bias}} \end{aligned} \tag{11}$$

This last equation can be written in a neural network form:

$$J_j(\mathbf{x}(t)) = h(t) * \left( \sum_i \omega_{ji} a_i(t) + \sum_k \omega_{jk} b_k(t) \right) + J_j^{\text{bias}} \tag{12}$$

where $\omega_{ji} = \alpha_j \left\langle \tilde{\phi}_j^{\mathbf{x}} \mathbf{A}' \phi_i^{\mathbf{x}} \right\rangle$ and $\omega_{jk} = \alpha_j \left\langle \tilde{\phi}_j^{\mathbf{x}} \mathbf{B}' \phi_k^{\mathbf{y}} \right\rangle$ are the recurrent and feedforward weights, respectively.

## 5   Efficient implementation of the SNN

In this section, we describe the two distinct steps carried out when implementing the SNN: creating and running the network. The first step has no computational constraints whereas the second must be very efficient in order to be successfully deployed in the closed-loop experimental setting.

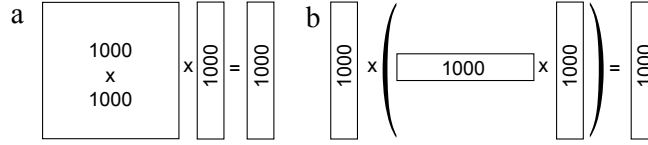

Figure 3: Computing a 1000-neuron pool's recurrent connections. **a.** Using connection weights requires multiplying a $1000 \times 1000$ matrix by a $1000 \times 1$ vector. **b.** Operating in the lower-dimensional state space requires multiplying a $1 \times 1000$ vector by a $1000 \times 1$ vector to get the decoded state, multiplying this state by a component of the $\mathbf{A}'$ matrix to update it, and multiplying the updated state by a $1000 \times 1$ vector to re-encode it as firing rates, which are then used to update the soma current for every neuron.

**Network creation:** This step generates, for a specified number of neurons composing the network, the gain $\alpha_j$, bias current $J_j^{\text{bias}}$, preferred direction $\tilde{\phi}_j^{\mathbf{x}}$, and decoding weight $\phi_j^{\mathbf{x}}$ for each neuron. The preferred directions $\tilde{\phi}_j^{\mathbf{x}}$ are drawn randomly from a uniform distribution over the unit sphere. The maximum firing rate, max $G(J_j(\mathbf{x}))$, and the normalized x-axis intercept, $G(J_j(\mathbf{x})) = 0$, are drawn randomly from a uniform distribution on [200, 400] Hz and [-1, 1], respectively. From these two specifications, $\alpha_j$ and $J_j^{\text{bias}}$ are computed using Equation 2 and Equation 3. The decoding weights $\phi_j^{\mathbf{x}}$ are computed by minimizing the mean square error (Equation 6).

For efficient implementation, we used two 1D integrators (i.e., two recurrent neuron pools, with each pool representing a scalar) rather than a single 3D integrator (i.e., one recurrent neuron pool, with the pool representing a 3D vector by itself) [13]. The constant 1 is fed to the 1D integrators as an input, rather than continuously integrated as part of the state vector. We also replaced the $b_k(t)$ units' spike rates (Figure 1, *middle*) with the 192 neural measurements (spike counts in 50ms bins), which is equivalent to choosing $\phi_k^{\mathbf{y}}$ from a standard basis (i.e., a unit vector with 1 at the $k^{\text{th}}$ position and 0 everywhere else) [7].

**Network simulation:** This step runs the simulation to update the soma current for every neuron, based on input spikes. The soma voltage is then updated following RC circuit dynamics. Gaussian noise is normally added at this step, the rest of the simulation being noiseless. Neurons with soma voltage above threshold generate a spike and enter their refractory period. The neuron firing rates are decoded using the linear decoding weights to get the updated states values, $x$ and $y$-velocity. These values are smoothed with a filter identical to $h(t)$, but with $\tau$ set to 5ms instead of 20ms to avoid introducing significant delay. Then the simulation step starts over again.

In order to ensure rapid execution of the simulation step, neuron interactions are not updated directly using the connection matrix (Equation 12), but rather indirectly with the decoding matrix $\phi_j^{\mathbf{x}}$, dynamics matrix $\mathbf{A}'$, and preferred direction matrix $\tilde{\phi}_j^{\mathbf{x}}$ (Equation 11). To see why this is more efficient, suppose we have 1000 neurons in the $a$ population for each of the state vector's two scalars. Computing the recurrent connections using connection weights requires multiplying a $1000 \times 1000$ matrix by a 1000-dimensional vector (Figure 3a). This requires $10^6$ multiplications and about $10^6$ sums. Decoding each scalar (i.e., $\sum_i a_i(t)\phi_i^{\mathbf{x}}$), however, requires only 1000 multiplications and 1000 sums. The decoded state vector is then updated by multiplying it by the (diagonal) $\mathbf{A}'$ matrix, another 2 products and 1 sum. The updated state vector is then encoded by multiplying it with the neurons' preferred direction vectors, another 1000 multiplications per scalar (Figure 3b). The resulting total of about 3000 operations is nearly three orders of magnitude fewer than using the connection weights to compute the identical transformation.

To measure the speedup, we simulated a 2,000-neuron network on a computer running Matlab 2011a (Intel Core i7, 2.7-GHz, Mac OS X Lion). Although the exact run-times depend on the computing hardware and software, the run-time reduction factor should remain approximately constant across platforms. For each reported result, we ran the simulation 10 times to obtain a reliable estimate of the execution time. The run-time for neuron interactions using the recurrent connection weights was 9.9ms and dropped to $2.7\mu$s in the lower-dimensional space, approximately a 3,500-fold speedup. Only the recurrent interactions benefit from the speedup, the execution time for the rest of the operations remaining constant. The run-time for a 50ms network simulation using the recurrent connec-

Table 1: Model parameters

| Symbol | Range | Description |
|---|---|---|
| max $G(J_j(\mathbf{x}))$ | 200-400 Hz | Maximum firing rate |
| $G(J_j(\mathbf{x})) = 0$ | $-1$ to 1 | Normalized x-axis intercept |
| $J_j^{\text{bias}}$ | Satisfies first two | Bias current |
| $\alpha_j$ | Satisfies first two | Gain factor |
| $\tilde{\phi}_j^{\mathbf{x}}$ | $\left\|\tilde{\phi}_j^{\mathbf{x}}\right\| = 1$ | Preferred-direction vector |
| $\sigma^2$ | 0.1 | Gaussian noise variance |
| $\tau_j^{\text{RC}}$ | 20 ms | RC time constant |
| $\tau_j^{\text{ref}}$ | 1 ms | Refractory period |
| $\tau_j^{\text{PSC}}$ | 20 ms | PSC time constant |

tion weights was 0.94s and dropped to 0.0198s in the lower-dimensional space, a 47-fold speedup. These results demonstrate the efficiency the lower-dimensional space offers, which made the closed-loop application of SNNs possible.

## 6 Closed-loop implementation

An adult male rhesus macaque (monkey J) was trained to perform a center-out-and-back reaching task for juice rewards to one of eight targets, with a 500ms hold time (Figure 4a) [1]. All animal protocols and procedures were approved by the Stanford Institutional Animal Care and Use Committee. Hand position was measured using a Polaris optical tracking system at 60Hz (Northern Digital Inc.). Neural data were recorded from two 96-electrode silicon arrays (Blackrock Microsystems) implanted in the dorsal pre-motor and motor cortex. These recordings (-4.5 RMS threshold crossing applied to each electrode's signal) yielded tuned activity for the direction and speed of arm movements. As detailed in [1], a standard Kalman filter model was fit by correlating the observed hand kinematics with the simultaneously measured neural signals, while the monkey moved his arm to acquire virtual targets (Figure 2). The resulting model was used in a closed-loop system to control an on-screen cursor in real-time (Figure 4a, Decoder block). A steady-state version of this model serves as the standard against which the SNN implementation's performance is compared.

We built a SNN using the NEF methodology based on derived Kalman filter parameters mentioned above. This SNN was then simulated on an xPC Target (Mathworks) x86 system (Dell T3400, Intel Core 2 Duo E8600, 3.33GHz). It ran in closed-loop, replacing the standard Kalman filter as the decoder block in Figure 4a. The parameter values listed in Table 1 were used for the SNN implementation. We ensured that the time constants $\tau_i^{\text{RC}}, \tau_i^{\text{ref}}$, and $\tau_i^{\text{PSC}}$ were smaller than the implementation's time step (50ms). Noise was not explicitly added. It arose naturally from the fluctuations produced by representing a scalar with filtered spike trains, which has been shown to have effects similar to Gaussian noise [11]. For the purpose of computing the linear decoding weights (i.e., $\Gamma$), we modeled the resulting noise as Gaussian with a variance of 0.1.

A 2,000-neuron version of the SNN-based decoder was tested in a closed-loop system, the largest network our embedded MatLab implementation could run in real-time. There were 1206 trials total among which 301 (center-outs only) were performed with the SNN and 302 with the standard (steady-state) Kalman filter. The block structure was randomized and interleaved, so that there is no behavioral bias present in the findings. 100 trials under hand control are used as a baseline comparison. Success corresponds to a target acquisition under 1500ms, with 500ms hold time. Success rates were higher than 99% on all blocks for the SNN implementation and 100% for the standard Kalman filter. The average time to acquire the target was slightly slower for the SNN (Figure 5b)—711ms vs. 661ms, respectively—we believe this could be improved by using more neurons in the SNN.[1] The average distance to target (Figure 5a) and the average velocity of the cursor (Figure 5c) are very similar.

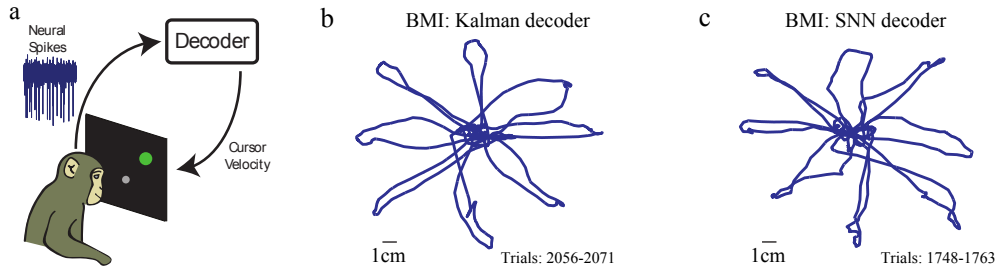

Figure 4: Experimental setup and results. **a.** Data are recorded from two 96-channel silicon electrode arrays implanted in dorsal pre-motor and motor cortex of an adult male monkey performing a center-out-and-back reach task for juice rewards to one of eight targets with a 500ms hold time. **b.** BMI position kinematics of 16 continuous trials for the standard Kalman filter implementation. **c.** BMI position kinematics of 16 continuous trials for the SNN implementation.

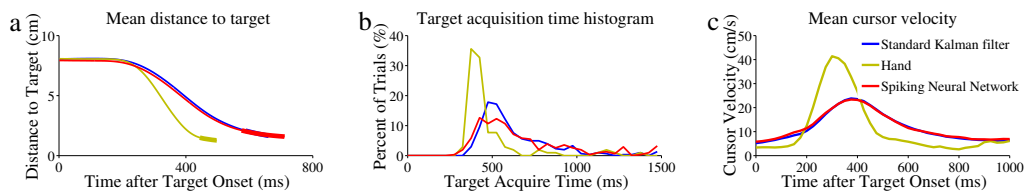

Figure 5: SNN (red) performance compared to standard Kalman filter (blue) (hand trials are shown for reference (yellow)). The SNN achieves similar results—success rates are higher than 99% on all blocks—as the standard Kalman filter implementation. **a.** Plot of distance to target vs. time both after target onset for different control modalities. The thicker traces represent the average time when the cursor first enters the acceptance window until successfully entering for the 500ms hold time. **b.** Histogram of target acquisition time. **c.** Plot of mean cursor velocity vs. time.

## 7 Conclusions and future work

The SNN's performance was quite comparable to that produced by a standard Kalman filter implementation. The 2,000-neuron network had success rates higher than 99% on all blocks, with mean distance to target, target acquisition time, and mean cursor velocity curves very similar to the ones obtained with the standard implementation. Future work will explore whether these results extend to additional animals. As the Kalman filter and its variants are the state-of-the-art in cortically-controlled motor prostheses [1]-[5], these simulations provide confidence that similar levels of performance can be attained with a neuromorphic system, which can potentially overcome the power constraints set by clinical applications.

Our ultimate goal is to develop an ultra-low-power neuromorphic chip for prosthetic applications on to which control theory algorithms can be mapped using the NEF. As our next step in this direction, we will begin exploring this mapping with *Neurogrid*, a hardware platform with sixteen programmable neuromorphic chips that can simulate up to a million spiking neurons in real-time [9]. However, bandwidth limitations prevent Neurogrid from realizing random connectivity patterns. It can only connect each neuron to thousands of others if neighboring neurons share common inputs — just as they do in the cortex. Such columnar organization may be possible with NEF-generated networks if preferred directions vectors are assigned topographically rather than randomly. Implementing this constraint effectively is a subject of ongoing research.

**Acknowledgment**

This work was supported in part by the Belgian American Education Foundation(J. Dethier), Stanford NIH Medical Scientist Training Program (MSTP) and Soros Fellowship (P. Nuyujukian), DARPA Revolutionizing Prosthetics program (N66001-06-C-8005, K. V. Shenoy), and two NIH Director's Pioneer Awards (DP1-OD006409, K. V. Shenoy; DPI-OD000965, K. Boahen).

## Footnotes

[1] Off-line, the SNN performed better as we increased the number of neurons [7].

# References

[1] V. Gilja, Towards clinically viable neural prosthetic systems, *Ph.D. Thesis, Department of Computer Science, Stanford University*, 2010, pp 19–22 and pp 57–73.

[2] V. Gilja, P. Nuyujukian, C.A. Chestek, J.P. Cunningham, J.M. Fan, B.M. Yu, S.I. Ryu, and K.V. Shenoy, A high-performance continuous cortically-controlled prosthesis enabled by feedback control design, *2010 Neuroscience Meeting Planner*, San Diego, CA: Society for Neuroscience, 2010.

[3] P. Nuyujukian, V. Gilja, C.A. Chestek, J.P. Cunningham, J.M. Fan, B.M. Yu, S.I. Ryu, and K.V. Shenoy, Generalization and robustness of a continuous cortically-controlled prosthesis enabled by feedback control design, *2010 Neuroscience Meeting Planner*, San Diego, CA: Society for Neuroscience, 2010.

[4] V. Gilja, C.A. Chestek, I. Diester, J.M. Henderson, K. Deisseroth, and K.V. Shenoy, Challenges and opportunities for next-generation intra-cortically based neural prostheses, *IEEE Transactions on Biomedical Engineering*, 2011, in press.

[5] S.P. Kim, J.D. Simeral, L.R. Hochberg, J.P. Donoghue, and M.J. Black, Neural control of computer cursor velocity by decoding motor cortical spiking activity in humans with tetraplegia, *Journal of Neural Engineering*, vol. 5, 2008, pp 455–476.

[6] S. Kim, P. Tathireddy, R.A. Normann, and F. Solzbacher, Thermal impact of an active 3-D microelectrode array implanted in the brain, *IEEE Transactions on Neural Systems and Rehabilitation Engineering*, vol. 15, 2007, pp 493–501.

[7] J. Dethier, V. Gilja, P. Nuyujukian, S.A. Elassaad, K.V. Shenoy, and K. Boahen, Spiking neural network decoder for brain-machine interfaces, *IEEE Engineering in Medicine & Biology Society Conference on Neural Engineering*, Cancun, Mexico, 2011, pp 396–399.

[8] K. Boahen, Neuromorphic microchips, *Scientific American*, vol. 292(5), 2005, pp 56–63.

[9] R. Silver, K. Boahen, S. Grillner, N. Kopell, and K.L. Olsen, Neurotech for neuroscience: unifying concepts, organizing principles, and emerging tools, *Journal of Neuroscience*, vol. 27(44), 2007, pp 11807–11819.

[10] J.V. Arthur and K. Boahen, Silicon neuron design: the dynamical systems approach, *IEEE Transactions on Circuits and Systems*, vol. 58(5), 2011, pp 1034-1043.

[11] C. Eliasmith and C.H. Anderson, *Neural engineering: computation, representation, and dynamics in neurobiological systems*, MIT Press, Cambridge, MA; 2003.

[12] C. Eliasmith, A unified approach to building and controlling spiking attractor networks, *Neural Computation*, vol. 17, 2005, pp 1276–1314.

[13] R. Singh and C. Eliasmith, Higher-dimensional neurons explain the tuning and dynamics of working memory cells, *The Journal of Neuroscience*, vol. 26(14), 2006, pp 3667–3678.

[14] C. Eliasmith, How to build a brain: from function to implementation, *Synthese*, vol. 159(3), 2007, pp 373–388.

[15] R.E. Kalman, A new approach to linear filtering and prediction problems, *Transactions of the ASME–Journal of Basic Engineering*, vol. 82(Series D), 1960, pp 35–45.

[16] G. Welsh and G. Bishop, An introduction to the Kalman Filter, *University of North Carolina at Chapel Hill Chapel Hill NC*, vol. 95(TR 95-041), 1995, pp 1–16.

[17] W.Q. Malik, W. Truccolo, E.N. Brown, and L.R. Hochberg, Efficient decoding with steady-state Kalman filter in neural interface systems, *IEEE Transactions on Neural Systems and Rehabilitation Engineering*, vol. 19(1), 2011, pp 25–34.

